# Semi-supervised Learning with Penalized Probabilistic Clustering

**Zhengdong Lu and Todd K. Leen**
Department of Computer Science and Engineering
OGI School of Science and Engineering , OHSU
Beaverton, OR 97006
{zhengdon,tleen}@cse.ogi.edu

## Abstract

While clustering is usually an unsupervised operation, there are circumstances in which we believe (with varying degrees of certainty) that items A and B should be assigned to the same cluster, while items A and C should not. We would like such *pairwise relations* to influence cluster assignments of out-of-sample data in a manner consistent with the prior knowledge expressed in the training set. Our starting point is probabilistic clustering based on Gaussian mixture models (GMM) of the data distribution. We express clustering preferences in the prior distribution over assignments of data points to clusters. This prior penalizes cluster assignments according to the degree with which they violate the preferences. We fit the model parameters with EM. Experiments on a variety of data sets show that PPC can consistently improve clustering results.

## 1 Introduction

While clustering is usually executed completely unsupervised, there are circumstances in which we have prior belief that pairs of samples should (or should not) be assigned to the same cluster. Such *pairwise relations* may arise from a perceived similarity (or dissimilarity) between samples, or from a desire that the algorithmically generated clusters match the geometric cluster structure perceived by the experimenter in the original data. Continuity, which suggests that neighboring pairs of samples in a time series or in an image are likely to belong to the same class of object, is also a source of clustering preferences. We would like these preferences to be incorporated into the cluster structure so that the assignment of out-of-sample data to clusters captures the concept(s) that give rise to the preferences expressed in the training data.

Some work [1, 2, 3] has been done on adopting traditional clustering methods, such as K-means, to incorporate pairwise relations. These models are based on hard clustering and the clustering preferences are expressed as *hard pairwise constraints* that *must* be satisfied. While this work was in progress, we became aware of the algorithm of Shental *et al.* [4] who propose a Gaussian mixture model (GMM) for clustering that incorporates hard pairwise constraints.

In this paper, we propose a soft clustering algorithm based on GMM that expresses cluster-

ing preferences (in the form of pairwise relations) in the *prior probability on assignments of data points to clusters*. This framework naturally accommodates both *hard constraints* and *soft preferences* in a framework in which the preferences are expressed as a Bayesian probability that pairs of points should (or should not) be assigned to the same cluster. We call the algorithm Penalized Probabilistic Clustering (PPC). Experiments on several datasets demonstrate that PPC can consistently improve the clustering result by incorporating reliable prior knowledge.

## 2 Prior Knowledge for Cluster Assignments

PPC begins with a standard GMM

$$P(x|\Theta) = \sum_{\alpha=1}^{M} \pi_\alpha P(x|\alpha, \theta_\alpha)$$

where $\Theta = (\pi_1, \ldots \pi_K, \theta_1, \ldots, \theta_K)$. We augment the dataset $X = \{x_i\}$, $i = 1 \ldots N$ with (latent) cluster assignments $Z = z(x_i), i = 1, \ldots, N$ to form the familiar *complete data* $(X, Z)$. The complete data likelihood is

$$P(X, Z|\Theta) = P(X|Z, \Theta)P(Z|\Theta). \tag{1}$$

### 2.1 Prior distribution in latent space

We incorporate our clustering preferences by manipulating the *prior probability $P(Z|\Theta)$*. In the standard Gaussian mixture model, the prior distribution is trivial: $P(Z|\Theta) = \prod_i \pi_{z_i}$. We incorporate prior knowledge (our clustering preferences) through a weighting function $g(Z)$ that has large values when the assignment of data points to clusters $Z$ conforms to our preferences, and low values when $Z$ conflicts with our preferences. Hence we write

$$P(Z|\Theta, G) = \frac{\prod_i \pi_{z_i} g(Z)}{\sum_Z \prod_j \pi_{z_j} g(Z)} \equiv \frac{1}{K} \prod_i \pi_{z_i} g(Z) \tag{2}$$

where the sum is over all possible assignments of the data to clusters. The likelihood of the data, *given a* specific cluster assignment, is independent of the cluster assignment preferences, and so the complete data likelihood is

$$P(X, Z|\Theta, G) = P(X|Z, \Theta)\frac{1}{K} \prod_i \pi_{z_i} g(Z) = \frac{1}{K} P(X, Z|\Theta)g(Z), \tag{3}$$

where $P(X, Z|\Theta)$ is the complete data likelihood for a *standard* GMM. The data likelihood is the sum of complete data likelihood over all possible $Z$, that is, $L(X|\Theta) = P(X|\Theta, G) = \sum_Z P(X, Z|\Theta, G)$, which can be maximized with the EM algorithm. Once the model parameters are fit, we do soft clustering according to the posterior probabilities for new data $p(\alpha|x, \Theta)$. (Note that cluster assignment preferences are *not* expressed for the new data, only for the training data.)

### 2.2 Pairwise relations

Pairwise relations provide a special case of the framework discussed above. We specify two types of pairwise relations:

- **link**: two sample should be assigned into one cluster
- **do-not-link**: two samples should be assigned into different clusters.

The weighting factor given to the cluster assignment configuration $Z$ is simple:

$$g(Z) = \prod_{i,j} \exp(W_{ij}^p \, \delta(z_i, z_j)),$$

where $\delta$ is the Kronecker $\delta$-function and $W_{ij}^p$ is the weight associated with sample pair $(x_i, x_j)$. It satisfies

$$W_{ij}^p \in [-\infty, \infty], \ W_{ij}^p = W_{ji}^p.$$

The weight $W_{ij}^p$ reflects our preference and confidence in assigning $x_i$ and $x_j$ into one cluster. We use a positive $W_{ij}^p$ when we prefer to assign $x_i$ and $x_j$ into one cluster (link), and a negative $W_{ij}^p$ when we prefer to assign them into different clusters (do-not-link). The value $|W_{ij}^p|$ reflects how certain we are in the preference. If $W_{ij}^p = 0$, we have no prior knowledge on the assignment relevancy of $x_i$ and $x_j$. In the extreme cases where $|W_{ij}^p| \to \infty$, the $Z$ violating the pairwise relations about $x_i$ and $x_j$ have zero prior probability, since for those assignments

$$P(Z|\Theta, G) = \frac{\prod_n \pi_{z_n} \prod_{i,j} \exp(W_{ij}^p \, \delta(z_i, z_j))}{\sum_Z \prod_n \pi_{z_n} \prod_{i,j} \exp(W_{ij}^p \, \delta(z_i, z_j))} \to 0.$$

Then the relations become *hard constraints*, while the relations with $|W_{ij}^p| < \infty$ are called *soft preferences*. In the remainder of this paper, we will use $W^p$ to denote the prior knowledge on pairwise relations, that is

$$P(X, Z|\Theta, W^p) = \frac{1}{K} P(X, Z|\Theta) \prod_{i,j} \exp(W_{ij}^p \, \delta(z_i, z_j)) \qquad (4)$$

## 2.3 Model fitting

We use the EM algorithm [5] to fit the model parameters $\Theta$.

$$\Theta^* = \arg\max_{\Theta} L(X|\Theta, G)$$

The expectation step (E-step) and maximization step (M-step) are

$$\texttt{E-step:} \quad Q(\Theta, \Theta^{(t-1)}) = E_{Z|X}(\log P(X, Z|\Theta, G)|X, \Theta^{(t-1)}, G)$$

$$\texttt{M-step:} \quad \Theta^{(t)} = \arg\max_{\Theta} Q(\Theta, \Theta^{(t-1)})$$

In the M-step, the optimal mean and covariance matrix of each component is:

$$\mu_k = \frac{\sum_{j=1}^N x_j P(k|x_j, \Theta^{(t-1)}, G)}{\sum_{j=1}^N P(k|x_j, \Theta^{(t-1)}, G)}$$

$$\Sigma_k = \frac{\sum_{j=1}^N P(k|x_j, \Theta^{(t-1)}, G)(x_j - \mu_k)(x_j - \mu_k)^T}{\sum_{j=1}^N P(k|x_j, \Theta^{(t-1)}, G)}.$$

However, the update of prior probability of each component is more difficult than for the standard GMM, we need to find

$$\pi \equiv \{\pi_1, \ldots, \pi_m\} = \arg\max_{\pi} \sum_{l=1}^M \sum_{i=1}^N \log \pi_l P(l|x_i, \Theta^{(t-1)}, G) - \log K(\pi).$$

In this paper, we use a numerical method to find the solution.

### 2.4 Posterior Inference and Gibbs sampling

The M-step requires the cluster membership posterior. Computing this posterior is simple for the standard GMM since each data point $x_i$ can be assigned to a cluster independent of the other data points and we have the familiar cluster origin posterior $p(z_i = k|x_i, \Theta)$.

For the PPC model calculating the posteriors is no longer trivial. If two sample points, $x_i$ and $x_j$ participate in a pairwise relations, equation (4) tells us

$$P(z_i, z_j|X, \Theta, W^p) \neq P(z_i|X, \Theta, W^p)P(z_j|X, \Theta, W^p) \ .$$

and the posterior probability of $x_i$ and $x_j$ cannot be computed separately.

For pairwise relations, the joint posterior distribution must be calculated over the entire transitive closure of the "link" or "do-not-link" relations. See Fig. 1 for an illustration.

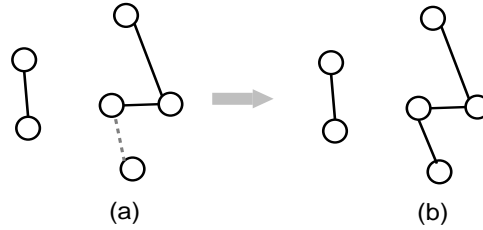

$$\qquad\qquad (a) \qquad\qquad\qquad (b)$$

Figure 1: (a) Links (solid line) and do-not-links (dotted line) among six samples; (b) Relevancy (solid line) translated from links in (a)

In the remainder of this paper, we will refer to the smallest sets of samples whose posterior assignment probabilities can be calculated independently as *cliques*. The posterior probability of a given sample $x_i$ in a clique $T$ is calculated by marginalizing the posterior over the entire clique

$$P(z_i = k|X, \Theta, W^p) = \sum_{Z_T|z_i=k} P(Z_T|X_T, \Theta, W^p),$$

with the posterior on the clique given by

$$P(Z_T|X_T, \Theta, W^p) = \frac{P(Z_T, X_T|\Theta, W^p)}{P(X_T|\Theta, W^p)} = \frac{P(Z_T, X_T|\Theta, W^p)}{\sum_{Z'_T} P(Z'_T, X_T|\Theta, W^p)}.$$

Computing the posterior probability of a sample in clique $T$ requires time complexity $O(M^{|T|})$, where $|T|$ is the size of clique $T$ and $M$ is the number of components in the mixture model. This is very expensive if $|T|$ is very big and model size $M \geq 2$. Hence small size cliques are required to make the marginalization computationally reasonable.

In some circumstances it is natural to limit ourselves to the special case of pairwise relation with $|T| \leq 2$, called *non-overlapping relations*. See Fig. 2 for illustration. More generally, we can avoid the expensive computation in posterior inference by breaking large clique into many small ones. To do this, we need to ignore some links or do-not-links. In section 3.2, we will give an application of this idea.

For some choices of $g(Z)$, the posterior probability can be given in a simple form even when the clique is big. One example is when there are only hard links. This case is useful when we are sure that a group of samples are from one source. For more general cases, where exact inference is computationally prohibitive, we propose to use Gibbs sampling [6] to estimate the posterior probability.

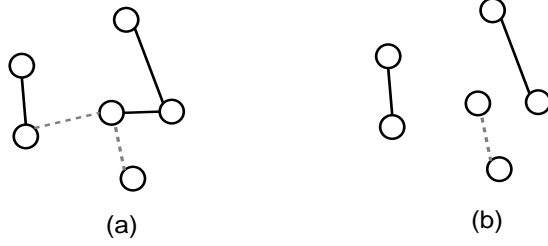

Figure 2: (a) Overlapping pairwise relations; (b) Non-overlapping pairwise relations.

In Gibbs sampling, we estimate $P(z_i|X, \Theta, G)$ as a sample mean

$$P(z_i = k|X, \Theta, G) = E(\delta(z_i, k)|X, \Theta, G) \approx \frac{1}{S} \sum_{t=1}^{S} \delta(z_i^{(t)}, k)$$

where the sum is over a sequence of $S$ samples from $P(Z|X, \Theta, G)$ generated by the Gibbs MCMC. The $t^{th}$ sample in the sequence is generated by the usual Gibbs sampling technique:

- Pick $z_1^{(t)}$ from distribution $P(z_1|z_2^{(t-1)}, z_3^{(t-1)}, ..., z_N^{(t-1)}, X, G, \Theta)$

- Pick $z_2^{(t)}$ from distribution $P(z_2|z_1^t, z_3^{(t-1)}, ..., z_N^{(t-1)}, X, G, \Theta)$

  $\cdots$

- Pick $z_N^{(t)}$ from distribution $P(z_N|z_1^{(t)}, z_2^{(t)}, ..., z_{N-1}^{(t)}, X, G, \Theta)$

For pairwise relations it is helpful to introduce some notation. Let $Z_{-i}$ denote an assignment of data points to clusters that leaves out the assignment of $x_i$. Let $U(i)$ be the indices of the set of samples that participate in a pairwise relation with sample $x_i$, $U(i) = \{j : W_{ij}^p \neq 0\}$. Then we have

$$P(z_i|Z_{-i}, X, \Theta, W^p) \propto P(x_i, z_i|\Theta) \prod_{j \in U(i)} \exp(2W_{ij}^p \, \delta(z_i, z_j)). \tag{5}$$

When $W^p$ is sparse, the size of $U(i)$ is small, thus calculating $P(z_i|Z_{-i}, X, \Theta, W^p)$ is very cheap and Gibbs sampling can effectively estimate the posterior probability.

## 3 Experiments

### 3.1 Clustering with different number of hard pairwise constraints

In this experiment, we demonstrate how the number of pairwise relations affects the performance of clustering. We apply PPC model to three UCI data sets: Iris,Waveform, and Pendigits. Iris data set has 150 samples and three classes, 50 samples in each class; Waveform data set has 5000 samples and three classes, $33\%$ samples in each class; Pendigits data set includes four classes (digits 0,6,8,9), each with 750 samples. All data sets have labels for all samples, which are used to generate the relations and to evaluate performance.

We try PPC (with component number same as the number of classes) with various number of pairwise relations. For each relations number, we conduct 100 runs and calculate the averaged classification accuracy. In each run, the data set is randomly split into training set ($90\%$) and test set ($10\%$). The pairwise relations are generated as follows: we randomly pick two samples from the *training* set without replacement and check their labels. If the two have the same label, we then add a link constraint between them; otherwise, we

add a do-not-link constraint. Note the generated pairwise relations are non-overlapping, as described in section 2.4. The model fitted on the training set is applied to test set. Experiment results on two data sets are shown in Fig. 3 (a) and (b) respectively. As Fig. 3 indicates, PPC can consistently improve its clustering accuracy on the training set when more pairwise constraints are added; also, the effect brought by constraints generalizes to the test set.

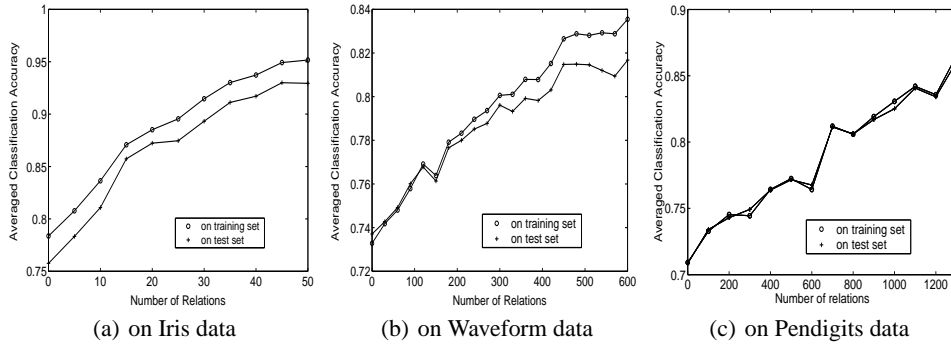

(a) on Iris data        (b) on Waveform data        (c) on Pendigits data

Figure 3: The performance of PPC with various number of relations

## 3.2 Hard pairwise constraints for encoding partial label

The experiment in this subsection shows the application of pairwise constraints on *partially labeled* data. For example, consider a problem with six classes $A, B, ..., F$. The classes are grouped into several class-sets $C_1 = \{A, B, C\}, C_2 = \{D, E\}, C_3 = \{F\}$. The samples are partially labeled in the sense that we are told which class-set a sample is from, but not which specific class it is from. We can logically derive a do-not-link constraint between any pair of samples known to belong to different class-sets, while no link constraint can be derived if each class-set has more than one class in it.

Fig. 4 (a) is a 120x400 region from Greenland ice sheet from NASA Langley DAAC. This region is partially labeled into snow area and non-snow area, as indicated in Fig. 4 (b). The snow area can be ice, melting snow or dry snow, while the non-snow area can be bare land, water or cloud. Each pixel has attributes from seven spectrum bands. To segment the image, we first divide the image into 5x5x7 blocks (175 dim vectors). We use the first 50 principal components as feature vectors.

For PPC, we use half of data samples for training set and the rest for test. Hard do-not-link constraints (only on training set) are generated as follows: for each block in the non-snow area, we randomly choose (without replacement) six blocks from the snow area to build do-not-link constraints. By doing this, we achieve cliques with size seven (1 non-snow block + 6 snow blocks). Like in section 3.1, we apply the model fitted with PPC to test set and combine the clustering results on both data sets into a complete picture. A typical clustering result of 3-component standard GMM and 3-component PPC are shown as Fig. 4 (c) and (d) respectively. From Fig. 4, standard GMM gives a clustering that is clearly in disagreement with the human labeling in Fig. 4 (b). The PPC segmentation makes far fewer mis-assignments of snow areas (tagged white and gray) to non-snow (black) than does the GMM. The PPC segmentation properly labels almost all of the non-snow regions as non-snow. Furthermore, the segmentation of the snow areas into the two classes (not labeled) tagged white and gray in Fig. 4 (d) reflects subtle differences in the snow regions captured by the gray-scale image from spectral channel 2, as shown in Fig. 4 (a).

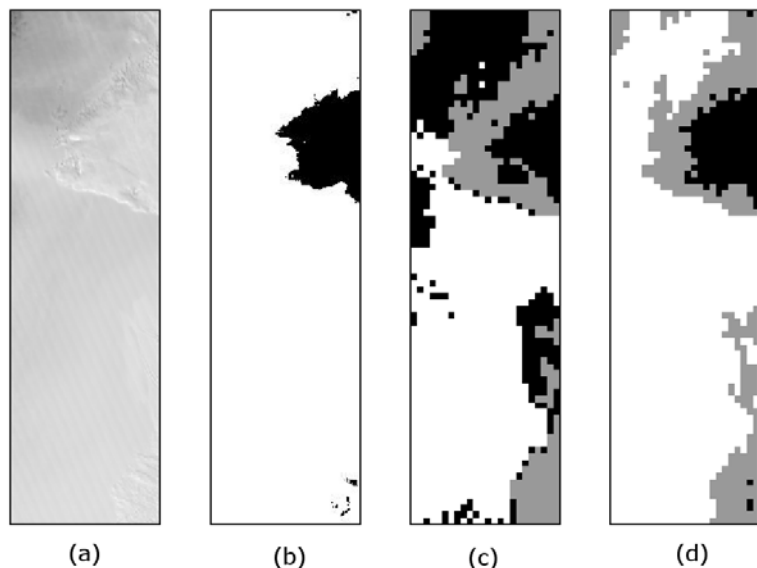

Figure 4: (a) Gray-scale image from the first spectral channel 2. (b) Partial label given by expert, black pixels denote non-snow area and white pixels denote snow area. Clustering result of standard GMM (c) and PPC (d). (c) and (d) are colored according to image blocks' assignment.

### 3.3 Soft pairwise preferences for texture image segmentation

In this subsection, we propose an unsupervised texture image segmentation algorithm as an application of PPC model. Like in section 3.2, the image is divided into blocks and rearranged into feature vectors. We use GMM to model those feature vectors, hoping each Gaussian component represents one texture. However, standard GMM often fails to give a good segmentation because it cannot make use of the spatial continuity of image, which is essential in many image segmentation models, such as random field [7]. In our algorithm, the spatial continuity is incorporated as the soft link preferences with uniform weight between each block and its neighbors. The *complete* data likelihood is

$$P(X, Z|\Theta, W^p) = \frac{1}{K} P(X, Z|\Theta) \prod_i \prod_{j \in U(i)} \exp(w\, \delta(z_i, z_j)), \qquad (6)$$

where $U(i)$ means the neighbors of the $i^{th}$ block. The EM algorithm can be roughly interpreted as iterating on two steps: 1) estimating the texture description (parameters of mixture model) based on segmentation, and 2) segmenting the image based on the texture description given by step 1. Gibbs sampling is used to estimate the posterior probability in each EM iteration. Equation (5) is reduced to

$$P(z_i|Z_{-i}, X, \Theta, W^p) \propto P(x_i, z_i|\Theta) \prod_{j \in U(i)} \exp(2w\, \delta(z_i, z_j)).$$

The image shown in Fig. 5 (a) is combined from four Brodatz textures [1] . This image is divided into 7x7 blocks and then rearranged to 49-dim vectors. We use those vectors' first five principal components as the associated feature vectors. For PPC model, the soft links

with weight $w$ are added between each block and its four neighbors, as shown in Fig. 5 (b). A typical clustering result of 4-component standard GMM and 4-component PPC with $w = 2$ are shown in Fig. 5 (c) and Fig. 5 (d) respectively. Obviously, PPC achieves a better segmentation after incorporating spatial continuity.

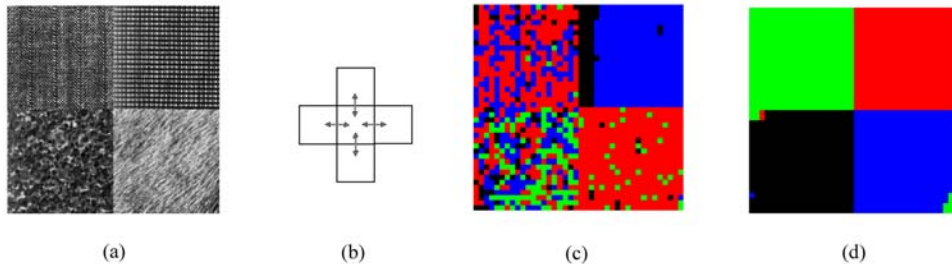

(a)            (b)            (c)            (d)

Figure 5: (a) Texture combination. (b) One block and its four neighbor. Clustering result of standard GMM (c) and PPC (d). (c) and (d) are shaded according to the blocks assignments to clusters.

## 4   Conclusion and Discussion

We have proposed a probabilistic clustering model that incorporates prior knowledge in the form of pairwise relations between samples. Unlike previous work in semi-supervised clustering, this work formulates clustering preferences as a Bayesian prior over the assignment of data points to clusters, and so naturally accommodates both hard constraints and soft preferences. For the computational difficulty brought by large cliques, we proposed a Markov chain estimation method to reduce the computational cost. Experiments on different data sets show that pairwise relations can consistently improve the performance of the clustering process.

### Acknowledgments

The authors thank Ashok Srivistava for helpful conversations. This work was funded by NASA Collaborative Agreement NCC 2-1264.

## Footnotes

[1]Downloaded from http://sipi.usc.edu/services/database/Database.html, April, 2004

## References

[1] K. Wagstaff, C. Cardie, S. Rogers, and S. Schroedl. Constrained K-means clustering with background knowledge. In *Proceedings of the Eighteenth International Conference on Machine Learning*, pages 577–584, 2001.

[2] S. Basu, A. Bannerjee, and R. Mooney. Semi-supervised clustering by seeding. In *Proceedings of the Nineteenth International Conference on Machine Learning*, pages 19–26, 2002.

[3] D. Klein, S. Kamvar, and C. Manning. From instance Level to space-level constraints: making the most of prior knowledge in data clustering. In *Proceedings of the Nineteenth International Conference on Machine Learning*, pages 307–313, 2002.

[4] N. Shental, A. Bar-Hillel, T. Hertz, and D. Weinshall. Computing Gaussian mixture models with EM using equivalence constraints. In *Advances in Neural Information Processing System*, volume 15, 2003.

[5] A. Dempster, N. Laird, and D. Rubin. Maximum likelihood from incomplete data via the EM algorithm. *Journal of the Royal Statistical Society, Series B*, 39:1–38, 1977.

[6] R. Neal. Probabilistic inference using Markov Chain Monte Carlo methods. Technical Report CRG-TR-93-1, Computer Science Department, Toronto University, 1993.

[7] C. Bouman and M. Shapiro. A multiscale random field model for Bayesian image segmentation. *IEEE Trans. Image Processing*, 3:162–177, March 1994.
